# Approximating Semidefinite Programs in Sublinear Time

**Dan Garber**
Technion - Israel Institute of Technology
Haifa 32000 Israel
dangar@cs.technion.ac.il

**Elad Hazan**
Technion - Israel Institute of Technology
Haifa 32000 Israel
ehazan@ie.technion.ac.il

## Abstract

In recent years semidefinite optimization has become a tool of major importance in various optimization and machine learning problems. In many of these problems the amount of data in practice is so large that there is a constant need for faster algorithms. In this work we present the first sublinear time approximation algorithm for semidefinite programs which we believe may be useful for such problems in which the size of data may cause even linear time algorithms to have prohibitive running times in practice. We present the algorithm and its analysis alongside with some theoretical lower bounds and an improved algorithm for the special problem of supervised learning of a distance metric.

## 1 Introduction

Semidefinite programming (SDP) has become a tool of great importance in optimization in the past years. In the field of combinatorial optimization for example, numerous approximation algorithms have been discovered starting with Goemans and Williamson [1] and [2, 3, 4]. In the field of machine learning solving semidefinite programs is at the heart of many learning tasks such as learning a distance metric [5], sparse PCA [6], multiple kernel learning [7], matrix completion [8], and more. It is often the case in machine learning that the data is assumed no be noisy and thus when considering the underlying optimization problem, one can settle for an approximated solution rather then an exact one. Moreover it is also common in such problems that the amounts of data are so large that fast approximation algorithms are preferable to exact generic solvers, such as interior-point methods, which have impractical running times and memory demands and are not scalable.

In the problem of learning a distance metric [5] one is given a set of points in $\mathbb{R}^n$ and similarity information in the form of pairs of points and a label indicating weather the two points are in the same class or not. The goal is to learn a distance metric over $\mathbb{R}^n$ which respects this similarity information. That is it assigns small distances to points in the same class and bigger distances to points in different classes. Learning such a metric is important for other learning tasks which rely on having a good metric over the input space, such as K-means, nearest-neighbours and kernel-based algorithms.

In this work we present the first approximation algorithm for general semidefinite programming which runs in time that is sublinear in the size of the input. For the special case of learning a pseudo-distance metric, we present an even faster sublinear time algorithm. Our algorithms are the fastest possible in terms of the number of constraints and the dimensionality, although slower than other methods in terms of the approximation guarantee.

### 1.1 Related Work

Semidefinite programming is a notoriously difficult optimization formulation, and has attracted a host of attempts at fast approximation methods. Klein and Lu [9] gave a fast approximate solver for

the MAX-CUT semidefinite relaxation of [1]. Various faster and more sophisticated approximate solvers followed [10, 11, 12], which feature near-linear running time albeit polynomial dependence on the approximation accuracy. For the special case of covering an packing SDP problems, [13] and [14] respectively give approximation algorithms with a smaller dependency on the approximation parameter $\epsilon$. Our algorithms are based on the recent work of [15] which described sublinear algorithms for various machine learning optimization problems such has linear classification and minimum enclosing ball. We describe here how such methods, coupled with techniques, may be used for semidefinite optimization.

## 2 Preliminaries

In this paper we denote vectors in $\mathbb{R}^n$ by a lower case letter (e.g. $v$) and matrices in $\mathbb{R}^{n \times n}$ by upper case letters (e.g. $A$). We denote by $\|v\|$ the standard euclidean norm of the vector $v$ and by $\|A\|$ the frobenius norm norm of the matrix $A$, that is $\|A\| = \sqrt{\sum_{i,j} A(i,j)^2}$. We denote by $\|v\|_1$ the $l_1$-norm of $v$. The notation $X \succeq 0$ states that the matrix $X$ is positive semi definite, i.e. it is symmetric and all of its eigenvalues are non negative. The notation $X \succeq B$ states that $X - B \succeq 0$. The notation $C \circ X$ is just the dot product between matrices, that is $C \circ X = \sum_{i,j} C(i,j)X(i,j)$. We denote by $\Delta_m$ the $m$-dimensional simplex, that is $\Delta_m = \{p | \sum_{i=1}^{m} p_i = 1, \forall i : p_i \geq 0\}$. We denote by $\mathbf{1}_n$ the all ones $n$-dimensional vector and by $\mathbf{0}_{n \times n}$ the all zeros $n \times n$ matrix. We denote by $I$ the identity matrix when its size is obvious from context. Throughout the paper we will use the complexity notation $\tilde{O}(\cdot)$ which is the same as the notation $O(\cdot)$ with the difference that it suppresses poly-logarithmic factors that depend on $n, m, \epsilon^{-1}$.

We consider the following general SDP problem

$$
\begin{aligned}
\text{Maximise} \quad & C \circ X \\
\text{subject to} \quad & A_i \circ X \;\geq\; 0 \quad i = 1, ..., m \\
& X \succeq 0
\end{aligned}
\tag{1}
$$

Where $C, A_1, ..., A_m \in \mathbb{R}^{n \times n}$. For reasons that will be made clearer in the analysis, we will assume that for all $i \in [m], \|A_i\| \leq 1$

The optimization problem (1) can be reduced to a feasibility problem by a standard reduction of performing a binary search over the value of the objective $C \circ X$ and adding an appropriate constraint. Thus we will only consider the feasibility problem of finding a solution that satisfies all constraints. The feasibility problem can be rewritten using the following min-max formulation

$$
\max_{X \succeq 0} \min_{i \in [m]} A_i \circ X
\tag{2}
$$

Clearly if the optimum value of (2) is non-negative, then a feasible solution exists and vice versa. Denoting the optimum of (2) by $\sigma$, an $\epsilon$ additive approximation algorithm to (2) is an algorithm that produces a solution $X$ such that $X \succeq 0$ and for all $i \in [m], A_i \circ X \geq \sigma - \epsilon$.

For the simplicity of the presentation we will only consider constraints of the form $A \circ X \geq 0$ but we mention in passing that SDPs with other linear constraints can be easily rewritten in the form of (1).

We will be interested in a solution to (2) which lies in the bounded semidefinite cone $\mathcal{K} = \{X | X \succeq 0, \text{Tr}(X) \leq 1\}$. The demand on a solution to (2) to have bounded trace is due to the observation that in case $\sigma > 0$, any solution needs to be bounded or else the products $A_i \circ X$ could be made to be arbitrarily large.

**Learning distance pseudo metrics** In the problem of learning a distance metric from examples, we are given a set triplets $\mathcal{S} = \{\{x_i, x_i', y_i\}\}_{i=1}^{m}$ such that $x_i, x_i' \in \mathbb{R}^n$ and $y_i \in \{-1, 1\}$. A value $y_i = 1$ indicates that the vectors $x_i, x_i'$ are in the same class and a value $y_i = -1$ indicates that they are from different classes. Our goal is to learn a pseudo-metric over $\mathbb{R}^n$ which respects the example set. A pseudo-metric is a function $d : \mathbb{R} \times \mathbb{R} \to \mathbb{R}$, which satisfies three conditions: (i) $d(x, x') \geq 0$, (ii) $d(x, x') = d(x', x)$, and (iii) $d(x_1, x_2) + d(x_2, x_3) \geq d(x_1, x_3)$. We consider pseudo-metrics of the form $d_A(x, x') \equiv \sqrt{(x - x')^\top A(x - x')}$. Its easily verified that if $A \succeq 0$ then $d_A$ is indeed a pseudo-metric. A reasonable demand from a "good" pseudo metric is that it separates the examples

(assuming such a separation exists). That is we would like to have a matrix $A \succeq 0$ and a threshold value $b \in \mathbb{R}$ such that for all $\{x_i, x_i', y_i\} \in \mathcal{S}$ it will hold that,

$$
\begin{aligned}
(d_A(x_i - x_i'))^2 = (x_i - x_i')^\top A(x_i - x_i') &\leq & b - \sigma/2 & \qquad y_i = 1 \\
(d_A(x_i - x_i'))^2 = (x_i - x_i')^\top A(x_i - x_i') &\geq & b + \sigma/2 & \qquad y_i = -1
\end{aligned}
\tag{3}
$$

where $\sigma$ is the margin of separation which we would like to maximize. Denoting by $v_i = (x_i - x_i')$ for all $i \in [m]$, (3) can be summarized into the following formalism:

$$
y_i \left( b - v_i^\top A v_i \right) \geq \sigma
$$

Without loss of generality we can assume that $b = 1$ and derive the following optimization problem

$$
\max_{A \succeq 0} \min_{i \in [m]} y_i \left( 1 - v_i^\top A v_i \right)
\tag{4}
$$

## 3  Algorithm for General SDP

Our algorithm for general SDPs is based on the generic framework for constrained optimization problems that fit a max-min formulation, such as (2), presented in [15]. Noticing that $\min_{i \in [m]} A_i \circ X = \min_{p \in \Delta_m} \sum_{i \in [m]} p(i) A_i \circ X$, we can rewrite (2) in the following way

$$
\max_{x \in \mathcal{K}} \min_{p \in \Delta_m} p(i) A_i^\top x
\tag{5}
$$

Building on [15], we use an iterative primal-dual algorithm that simulates a repeated game between two online algorithms: one that wishes to maximize $\sum_{i \in [m]} p(i) A_i \circ X$ as a function of $X$ and the other that wishes to minimize $\sum_{i \in [m]} p(i) A_i \circ X$ as a function of $p$. If both algorithms achieve sublinear regret, then this framework is known to approximate max-min problems such as (5), in case a feasible solution exists [16].

The primal algorithm which controls $X$ is a gradient ascent algorithm that given $p$ adds to the current solution a vector in the direction of the gradient $\sum_{i \in [m]} p(i) A_i$. Instead of adding the exact gradient we actually only sample from it by adding $A_i$ with probability $p(i)$ (lines 5-6). The dual algorithm which controls $p$ is a variant of the well known multiplicative (or exponential) update rule for online optimization over the simplex which updates the weight $p(i)$ according to the product $A_i \circ X$ (line 11). Here we replace the exact computation of $A_i \circ X$ by employing the $l_2$-sampling technique used in [15] in order to estimate this quantity by viewing only a single entry of the matrix $A_i$ (line 9). An important property of this sampling procedure is that if $\|A_i\| \leq 1$, then $\mathbb{E}[\tilde{v}_t(i)^2] \leq 1$. Thus, we can estimate the product $A_i \circ X$ with constant variance, which is important for our analysis. A problem that arises with this estimation procedure is that it might yield unbounded values which do not fit well with the multiplicative weights analysis. Thus we use a clipping procedure $\mathrm{clip}(z, V) \equiv \min\{V, \max\{-V, Z\}\}$ to bound these estimations in a certain range (line 10). Clipping the samples yields unbiased estimators of the products $A_i \circ X$ but the analysis shows that this bias is not harmful.

The algorithm is required to generate a solution $X \in \mathcal{K}$. This constraint is enforced by performing a projection step onto the convex set $\mathcal{K}$ after each gradient improvement step of the primal online algorithm. A projection of a matrix $Y \in \mathbb{R}^{n \times n}$ onto $\mathcal{K}$ is given by $Y_p = \arg\min_{X \in \mathcal{K}} \|Y - X\|$. Unlike the algorithms in [15] that perform optimization over simple sets such as the euclidean unit ball which is trivial to project onto, projecting onto the bounded semidefinite cone is more complicated and usually requires to diagonalize the projected matrix (assuming it is symmetric). Instead, we show that one can settle for an approximated projection which is faster to compute (line 4). Such approximated projections could be computed by Hazan's algorithm for offline optimization over the bounded semidefinite cone, presented in [12]. Hazan's algorithm gives the following guarantee

**Lemma 3.1.** *Given a matrix $Y \in \mathbb{R}^{n \times n}$, $\epsilon > 0$, let $f(X) = -\|Y - X\|^2$ and denote $X^* = \arg\max_{X \in \mathcal{K}} f(X)$. Then Hazan's algorithm produces a solution $\tilde{X} \in \mathcal{K}$ of rank at most $\epsilon^{-1}$ such that $\|Y - \tilde{X}\|^2 - \|Y - X^*\|^2 \leq \epsilon$ in $O\left(\frac{n^2}{\epsilon^{1.5}}\right)$ time.*

We can now state the running time of our algorithm.

**Lemma 3.2.** *Algorithm SublinearSDP has running time $\tilde{O}\left(\frac{m}{\epsilon^2} + \frac{n^2}{\epsilon^5}\right)$.*

**Algorithm 1** SublinearSDP

---

1: Input: $\epsilon > 0$, $A_i \in \mathbb{R}^{n \times n}$ for $i \in [m]$.

2: Let $T \leftarrow 60^2 \epsilon^{-2} \log m$, $Y_1 \leftarrow 0_{n \times n}$, $w_1 \leftarrow 1_m$, $\eta \leftarrow \sqrt{\frac{\log m}{T}}$, $\epsilon_P \leftarrow \epsilon/2$.

3: **for** $t = 1$ to T **do**

4:     $p_t \leftarrow \frac{w_t}{\|w_t\|_1}$, $X_t \leftarrow$ ApproxProject$(Y_t, \epsilon_P^2)$.

5:     Choose $i_t \in [m]$ by $i_t \leftarrow i$ w.p. $p_t(i)$.

6:     $Y_{t+1} \leftarrow Y_t + \frac{1}{\sqrt{2T}} A_{i_t}$

7:     Choose $(j_t, l_t) \in [n] \times [n]$ by $(j_t, l_t) \leftarrow (j, l)$ w.p. $X_t(j,l)^2 / \|X_t\|^2$.

8:     **for** $i \in [m]$ **do**

9:       $\tilde{v}_t \leftarrow A_i(j_t, l_t) \|X_t\|^2 / X_t(j_t, l_t)$

10:      $v_t(i) \leftarrow$ clip$(\tilde{v}_t(i), 1/\eta)$

11:      $w_{t+1}(i) \leftarrow w_t(i)(1 - \eta v_t(i) + \eta^2 v_t(i)^2)$

12:     **end for**

13: **end for**

14: **return** $\bar{X} = \frac{1}{T} \sum_t X_t$

---

We also have the following lower bound.

**Theorem 3.3.** *Any algorithm which computes an $\epsilon$-approximation with probability at least $\frac{2}{3}$ to (2) has running time $\Omega\left(\frac{m}{\epsilon^2} + \frac{n^2}{\epsilon^2}\right)$.*

We note that while the dependency of our algorithm on the number of constraints $m$ is close to optimal (up to poly-logarithmic factors), there is a gap of $\tilde{O}(\epsilon^{-3})$ between the dependency of our algorithm on the size of the constraint matrices $n^2$ and the above lower bound. Here it is important to note that our lower bound does not reflect the computational effort in computing a general solution that is positive semidefinite which is in fact the computational bottleneck of our algorithm (due to the use of the projection procedure).

## 4  Analysis

We begin with the presentation of the Multiplicative Weights algorithm used in our algorithm.

**Definition 4.1.** *Consider a sequence of vectors $q_1, ..., q_T \in \mathbb{R}^m$. The Multiplicative Weights (MW) algorithm is as follows. Let $0 < \eta \in \mathbb{R}$, $w_1 \leftarrow 1_m$, and for $t \geq 1$,*

$$p_t \leftarrow w_t / \|w_t\|_1, \quad w_{t+1} \leftarrow w_t(i)(1 - \eta q_t(i) + \eta^2 q_t(i)^2)$$

The following lemma gives a bound on the regret of the MW algorithm, suitable for the case in which the losses are random variables with bounded variance.

**Lemma 4.2.** *The MW algorithm satisfies*

$$\sum_{t \in [T]} p_t^\top q_t \leq \min_{i \in [m]} \sum_{t \in [T]} \max\{q_t(i), -\frac{1}{\eta}\} + \frac{\log m}{\eta} + \eta \sum_{t \in [t]} p_t^\top q_t^2$$

The following lemma gives concentration bounds on our random variables from their expectations.

**Lemma 4.3.** *For $1/4 \geq \eta \geq \sqrt{\frac{\log m}{T}}$, with probability at least $1 - O(1/m)$, it holds that*

$(i) \quad \max_{i \in [m]} \sum_{t \in [T]} [v_t(i) - A_i \circ X_t] \leq 4\eta T \quad (ii) \quad \left| \sum_{t \in [T]} A_{i_t} \circ X_t - \sum_{t \in [T]} p_t^\top v_t \right| \leq 8\eta T$

The following Lemma gives a regret bound on the lazy gradient ascent algorithm used in our algorithm (line 6). For a proof see Lemma A.2 in [17].

**Lemma 4.4.** *Consider matrices $A_1, ..., A_T \in \mathbb{R}^{n \times n}$ such that for all $i \in [m]$ $\|A_i\| \leq 1$. Let $X_0 = \mathbf{0}_{n \times n}$ and for all $t \geq 1$ let $X_{t+1} = \arg\min_{X \in \mathcal{K}} \left\| \frac{1}{\sqrt{2T}} \sum_{\tau=1}^{t} A_\tau - X \right\|$ Then*

$$\max_{X \in \mathcal{K}} \sum_{t \in [T]} A_t \circ X - \sum_{t \in [T]} A_t \circ X_t \leq 2\sqrt{2T}$$

We are now ready to state the main theorem and prove it.

**Theorem 4.5** (Main Theorem). *With probability 1/2, the SublinearSDP algorithm returns an $\epsilon$-additive approximation to (5).*

*Proof.* At first assume that the projection onto the set $\mathcal{K}$ in line 4 is an exact projection and not an approximation and denote by $\tilde{X}_t$ the exact projection of $Y_t$. In this case, by lemma 4.4 we have

$$\max_{x \in \mathcal{K}} \sum_{t \in [T]} A_{i_t} \circ X - \sum_{t \in [T]} A_{i_t} \circ \tilde{X}_t \leq 2\sqrt{2T} \tag{6}$$

By the law of cosines and lemma 3.1 we have for every $t \in [T]$

$$\|X_t - \tilde{X}_t\|^2 \leq \|Y_t - X_t\|^2 - \|Y_t - \tilde{X}_t\|^2 \leq \epsilon_P^2 \tag{7}$$

Rewriting (6) we have

$$\max_{x \in \mathcal{K}} \sum_{t \in [T]} A_{i_t} \circ X - \sum_{t \in [T]} A_{i_t} \circ X_t - \sum_{t \in [T]} A_{i_t} \circ (\tilde{X}_t - X_t) \leq 2\sqrt{2T}$$

Using the Cauchy-Schwarz inequality, $\|A_{i_t}\| \leq 1$ and (7) we get

$$\max_{x \in \mathcal{K}} \sum_{t \in [T]} A_{i_t} \circ X - \sum_{t \in [T]} A_{i_t} \circ X_t \leq 2\sqrt{2T} + \sum_{t \in [T]} \|A_{i_t}\| \|\tilde{X}_t - X_t\| \leq 2\sqrt{2T} + T\epsilon_P$$

Rearranging and plugging $\max_{x \in \mathcal{K}} \min_{i \in [m]} A_i \circ X = \sigma$ we get

$$\sum_{t \in [T]} A_{i_t} \circ X_t \geq T\sigma - 2\sqrt{2T} - T\epsilon_P \tag{8}$$

Turning to the MW part of the algorithm, by the MW Regret Lemma 4.2, and using the clipping of $v_t(i)$ we have

$$\sum_{t \in [T]} p_t^\top v_t \leq \min_{i \in [i]} \sum_{t \in [t]} v_t(i) + (\log m)/\eta + \eta \sum_{t \in [T]} p_t^\top v_t^2$$

By Lemma 4.3, with high probability and for any $i \in [n]$,

$$\sum_{t \in [T]} v_t(i) \leq \sum_{t \in [T]} A_i \circ X_t + 4\eta T$$

Thus with high probability it holds that

$$\sum_{t \in [T]} p_t^\top v_t \leq \min_{i \in [i]} \sum_{t \in [t]} A_i \circ X_t + (\log m)/\eta + \eta \sum_{t \in [T]} p_t^\top v_t^2 + 4\eta T \tag{9}$$

Combining (8) and (9) we get

$$\min_{i \in [i]} \sum_{t \in [t]} A_i \circ X_t \geq -(\log m)/\eta - \eta \sum_{t \in [T]} p_t^\top v_t^2 - 4\eta T + T\sigma$$

$$-2\sqrt{2T} - \left| \sum_{t \in [T]} p_t^\top v_t - \sum_{t \in [T]} A_{i_t} \circ X_t \right| - T\epsilon_P$$

By a simple Markov inequality argument it holds that w.p. at least 3/4,

$$\sum_{t \in [T]} p_t^\top v_t^2 \leq 8T$$

Combined with lemma 4.3, we have w.p. at least $\frac{3}{4} - O(\frac{1}{n}) \geq \frac{1}{2}$

$$\min_{\substack{i \in [i] \\ t \in [t]}} \sum A_i \circ X_t \quad \geq \quad -(\log m)/\eta - 8\eta T - 4\eta T + T\sigma - 2\sqrt{2T} - 8\eta T - T\epsilon_P$$

$$\geq \quad T\sigma - \frac{\log m}{\eta} - 20\eta T - 2\sqrt{2T} - T\epsilon_P$$

Dividing through by $T$ and plugging in our choice for $\eta$ and $\epsilon_P$, we have $\min_{i \in [m]} A_i \circ \bar{X} \geq \sigma - \epsilon$ w.p. at least 1/2. $\qquad \square$

## 5  Application to Learning Pseudo-Metrics

As in the problem of general SDP, we can also rewrite (4) by replacing the $\min_{i \in [m]}$ objective with $\min_{p \in \Delta_m}$ and arrive at the following formalism,

$$\max_{A \succeq 0} \min_{p \in \Delta_m} y_i \left(1 - v_i^\top A v_i\right) \tag{10}$$

As we demanded a solution to general SDP to have bounded trace, here we demand that $\|A\| \leq 1$. Letting $v_i' = \begin{pmatrix} v_i \\ 1 \end{pmatrix}$ and defining the set of matrices $\mathcal{P} = \left\{ \begin{pmatrix} A & 0 \\ 0 & -1 \end{pmatrix} \mid A \succeq 0, \|A\| \leq 1 \right\}$, we can rewrite (10) in the following form.

$$\max_{A \in \mathcal{P}} \min_{p \in \Delta_m} -y_i v_i' v_i'^\top \circ A \tag{11}$$

In what comes next, we use the notation $A_i = -y_i v_i' v_i'$.
Since projecting a matrix onto the set $\mathcal{P}$ is as easy as projecting a matrix onto the set $\{A \succeq 0, \|A\| \leq 1\}$, we assume for the simplicity of the presentation that the set on which we optimize is indeed $\mathcal{P} = \{A \succeq 0, \|A\| \leq 1\}$.
We proceed with presenting a simpler algorithm for this problem than the one given for general SDP. The gradient of $y_i v_i' v_i'^\top \circ A$ with respect to $A$ is a symmetric rank one matrix and here we have the following useful fact that was previously stated in [18].

**Theorem 5.1.** *If $A \in \mathbb{R}^{n \times n}$ is positive semi definite, $v \in \mathbb{R}^n$ and $\alpha \in \mathbb{R}$ then the matrix $B = A + \alpha v v^\top$ has at most one negative eigenvalue.*

The proof is due to the eigenvalue Interlacing Theorem (see [19] pp. 94-97 and [20] page 412).
Thus after performing a gradient step improvement of the form $Y_{t+1} = X_t + \eta y_i v_i v_i^\top$, projecting $Y_{t+1}$ onto to the feasible set $\mathcal{P}$ comes down to the removal of at most one eigenvalue in case we subtracted a rank one matrix ($y_{i_t} = -1$) or normalizing the $l_2$ norm in case we added a rank one matrix ($y_{i_t} = 1$). Since in practice computing eigenvalues fast, using the Power or Lanczos methods, can be done only up to a desired approximation, in fact the resulting projection $X_{t+1}$ might not be positive semidefinite. Nevertheless, we show by care-full analysis that we can still settle for a single eigenvector computation in order to compute an approximated projection with the price that $X_{t+1} \succeq -\epsilon^3 I$. That is $X_{t+1}$ might be slightly outside of the positive semidefinite cone. The benefit is an algorithm with improved performance over the general SDP algorithm since far less eigenvalue computations are required than in Hazan's algorithm.
The projection to the set $\mathcal{P}$ is carried out in lines 7-11. In line 7 we check if $Y_{t+1}$ has a negative eigenvalue and if so, we compute the corresponding eigenvector in line 8 and remove it in line 9. In line 11 we normalize the $l_2$ norm of the solution. The procedure Sample$(A_i, X_t)$ will be detailed later on when we discuss the running time.

The following Lemma is a variant of Zinkevich's Online Gradient Ascent algorithm [21] suitable for the use of approximated projections when $X_t$ is not necessarily inside the set $\mathcal{P}$.

**Lemma 5.2.** *Consider a set of matrices $A_1, ..., A_T \in \mathbb{R}^{n \times n}$ such that $\|A_i\| \leq 1$. Let $X_0 = \mathbf{0}_{n \times n}$ and for all $t \geq 0$ let*

$$Y_{t+1} = X_t + \eta A_t, \quad \tilde{X}_{t+1} = \arg \min_{X \in \mathcal{P}} \|Y_{t+1} - X\|$$

---

**Algorithm 2** SublinearPseudoMetric

---

1: Input: $\epsilon > 0$, $A_i = y_i v_i v_i^\top \in \mathbb{R}^{n \times n}$ for $i \in [m]$.
2: Let $T \leftarrow 60^2 \epsilon^{-2} \log m$, $X_1 =\leftarrow 0_{n \times n}$, $w_1 \leftarrow 1_m$, $\eta \leftarrow \sqrt{\frac{\log m}{T}}$.
3: **for** $t = 1$ to T **do**
4:    $p_t \leftarrow \frac{w_t}{\|w_t\|_1}$.
5:    Choose $i_t \in [m]$ by $i_t \leftarrow i$ w.p. $p_t(i)$.
6:    $Y_{t+1} \leftarrow X_t + \sqrt{\frac{2}{T}} y_{i_t} v_{i_t} v_{i_t}^\top$
7:    **if** $y_i < 0$ and $\lambda_{min}(Y_{t+1}) < 0$ **then**
8:        $u \leftarrow \arg\min_{z:\|z\|=1} z^\top Y_{t+1} z$
9:        $Y_{t+1} = Y_{t+1} - \lambda u u^\top$
10:   **end if**
11:   $X_{t+1} \leftarrow \frac{Y_{t+1}}{\max\{1, \|Y_{t+1}\|\}}$
12:   **for** $i \in [m]$ **do**
13:       $v_t(i) \leftarrow \text{clip}(\text{Sample}(A_i, X_t), 1/\eta)$
14:       $w_{t+1}(i) \leftarrow w_t(i)(1 - \eta v_t(i) + \eta^2 v_t(i)^2)$
15:   **end for**
16: **end for**
17: **return** $\bar{X} = \frac{1}{T} \sum_t X_t$

---

and let $X_{t+1}$ be such that $\left\| \tilde{X}_{t+1} - X_{t+1} \right\| \leq \epsilon_d$. Then, for a proper choice of $\eta$ it holds that,

$$\max_{X \in \mathcal{P}} \sum_{t \in [T]} A_t \circ X - \sum_{t \in [T]} A_t \circ X_t \leq \sqrt{2T} + \frac{3}{2} \epsilon_d T^{3/2}$$

The following lemma states the connection between the precision used in eigenvalues approximation in lines 7-8, and the quality of the approximated projection.

**Lemma 5.3.** *Assume that on each iteration $t$ of the algorithm, the eigenvalue computation in line 7 is a $\delta = \frac{\epsilon_d}{4T^{1.5}}$ additive approximation of the smallest eigenvalue of $Y_{t+1}$ and let $\tilde{X}_t = \arg\min_{X \in \mathcal{P}} \|Y_t - X\|$. It holds that*

$$\|\tilde{X}_t - X_t\| \leq \epsilon_d$$

**Theorem 5.4.** *Algorithm SublinearPseudoMetric computes an $\epsilon$ additive approximation to (11) w.p. $1/2$.*

*Proof.* Combining lemmas 5.2, 5.3 we have,

$$\max_{X \in \mathcal{P}} \sum_{t \in [T]} A_t \circ X - \sum_{t \in [T]} A_t \circ X_t \leq \sqrt{2T} + \frac{3}{2} \epsilon_d T^{3/2}$$

Setting $\epsilon_d = \frac{2\epsilon_P}{3\sqrt{T}}$ where $\epsilon_P$ is the same as in theorem 4.5 yields,

$$\arg\max_{X \in \mathcal{P}} \sum_{t \in [T]} A_t \circ X - \sum_{t \in [T]} A_t \circ X_t \leq \sqrt{2T} + \epsilon_P T$$

The rest of the proof follows the same lines as theorem 4.5. $\square$

We move on to discus the time complexity of the algorithm. It is easily observed from the algorithm that for all $t \in [T]$, the matrix $X_t$ can be represented as the sum of $k_t \leq 2T$ symmetric rank-one matrices. That is $X_t$ is of the form $X_t = \sum_{i \in [k_t]} \alpha_i z_i z_i^\top$, $\|z_i\| = 1$ for all $i$. Thus instead of computing $X_t$ explicitly, we may represent it by the vectors $z_i$ and scalars $\alpha_i$. Denote by $\alpha$ the vector of length $k_t$ in which the ith entry is just $\alpha_i$, for some iteration $t \in [T]$. Since $\|X_t\| \leq 1$ it holds that $\|\alpha\| \leq 1$. The sampling procedure $\text{Sample}(A_i, X_t)$ in line 13, returns the value $\frac{A_i(j,l)\|\alpha\|^2}{z_k(j)z_k(l)\alpha_k}$ with probability $\frac{\alpha_k^2}{\|\alpha\|^2} \cdot (z_k(j)z_k(l))^2$. That is we first sample a vector $z_i$ according to

$\alpha$ and then we sample an entry $(j, l)$ according to the chosen vector $z_i$. It is easily observed that $\tilde{v}_t(i) = \text{Sample}(A_i, X_t)$ is an unbiased estimator of $A_i \circ X_t$. It also holds that:

$$
\begin{aligned}
\mathbb{E}[\tilde{v}_t(i)^2] &= \sum_{j \in [n], l \in [n], k \in [k_t]} \left( \frac{\alpha_k^2}{\|\alpha\|^2} \left( z_k(j) z_k(l) \right)^2 \cdot \frac{A_i(j, l)^2 \|\alpha\|^4}{(z_k(j) z_k(l))^2 \alpha_k^2} \right) \\
&= k_t \|\alpha\|^2 \|A_i\|^2 = \tilde{O}(\epsilon^{-2})
\end{aligned}
$$

Thus taking $\tilde{v}_t(i)$ to be the average of $\tilde{O}(\epsilon^{-2})$ i.i.d samples as described above yields an unbiased estimator of $A_i \cdot X_t$ with variance at most 1 as required for the analysis of our algorithm.

We can now state the running time of the algorithm.

**Lemma 5.5.** *Algorithm SublinearPseudoMetric can be implemented to run in time $\tilde{O}\left( \frac{m}{\epsilon^4} + \frac{n}{\epsilon^{6.5}} \right)$.*

*Proof.* According the lemmas 5.3, 5.4, the required precision in eigenvalue approximation is $\frac{\epsilon}{O(1)T^2}$. Using the Lanczos method for eigenvalue approximation and the sparse representation of $X_t$ described above, a single eigenvalue computation takes $\tilde{O}(n\epsilon^{-4.5})$ time per iteration. Estimating the products $A_i \circ X_t$ on each iteration takes by the discussion above $\tilde{O}(m\epsilon^{-2})$. Overall the running time on all iteration is as stated in the lemma. $\square$

## 6  Conclusions

We have presented the first sublinear time algorithm for approximate semi-definite programming, a widely used optimization framework in machine learning. The algorithm's running time is optimal up to poly-logarithmic factors and its dependence on $\varepsilon$ - the approximation guarantee. The algorithm is based on the primal-dual approach of [15], and incorporates methods from previous SDP solvers [12].

For the problem of learning peudo-metrics, we have presented further improvements to the basic method which entail an algorithm that performs $O(\frac{\log n}{\varepsilon^2})$ iterations, each encompassing at most one approximate eigenvector computation.

### Acknowledgements

This work was supported in part by the IST Programme of the European Community, under the PASCAL2 Network of Excellence, IST-2007-216886. This publication only reflects the authors' views.

## References

[1] Michel. X. Goemans and David P. Williamson. Improved approximation algorithms for maximum cut and satisfiability problems using semidefinite programming. In *Journal of the ACM*, volume 42, pages 1115–1145, 1995.

[2] Sanjeev Arora, Satish Rao, and Umesh Vazirani. Expander flows, geometric embeddings and graph partitioning. In *Proceedings of the thirty-sixth annual ACM symposium on Theory of computing*, STOC '04, pages 222–231, 2004.

[3] Amit Agarwal, Moses Charikar, Konstantin Makarychev, and Yury Makarychev. O(sqrt(log n)) approximation algorithms for min uncut, min 2cnf deletion, and directed cut problems. In *Proceedings of the thirty-seventh annual ACM symposium on Theory of computing*, STOC '05, pages 573–581, 2005.

[4] Sanjeev Arora, James R. Lee, and Assaf Naor. Euclidean distortion and the sparsest cut. In *Proceedings of the thirty-seventh annual ACM symposium on Theory of computing*, STOC '05, pages 553–562, 2005.

[5] Eric P. Xing, Andrew Y. Ng, Michael I. Jordan, and Stuart Russell. Distance metric learning, with application to clustering with side-information. In *Advances in Neural Information Processing Systems 15*, pages 505–512, 2002.

[6] Alexandre d'Aspremont, Laurent El Ghaoui, Michael I. Jordan, and Gert R. G. Lanckriet. A direct formulation of sparse PCA using semidefinite programming. In *SIAM Review*, volume 49, pages 41–48, 2004.

[7] Gert R. G. Lanckriet, Nello Cristianini, Laurent El Ghaoui, Peter Bartlett, and Michael I. Jordan. Learning the kernel matrix with semi-definite programming. In *Journal of Machine Learning Research*, pages 27–72, 2004.

[8] Stephen Boyd and Lieven Vandenberghe. *Convex Optimization*. Cambridge University Press, 2004.

[9] Philip Klein and Hsueh-I Lu. Efficient approximation algorithms for semidefinite programs arising from max cut and coloring. In *Proceedings of the twenty-eighth annual ACM symposium on Theory of computing*, STOC '96, pages 338–347, 1996.

[10] Sanjeev Arora, Elad Hazan, and Satyen Kale. Fast algorithms for approximate semide.nite programming using the multiplicative weights update method. In *Proceedings of the 46th Annual IEEE Symposium on Foundations of Computer Science*, FOCS '05, pages 339–348, 2005.

[11] Sanjeev Arora and Satyen Kale. A combinatorial, primal-dual approach to semidefinite programs. In *Proceedings of the thirty-ninth annual ACM symposium on Theory of computing*, STOC '07, pages 227–236, 2007.

[12] Elad Hazan. Sparse approximate solutions to semidefinite programs. In *Proceedings of the 8th Latin American conference on Theoretical informatics*, LATIN'08, pages 306–316, 2008.

[13] Garud Iyengar, David J. Phillips, and Clifford Stein. Feasible and accurate algorithms for covering semidefinite programs. In *SWAT*, pages 150–162, 2010.

[14] Garud Iyengar, David J. Phillips, and Clifford Stein. Approximating semidefinite packing programs. In *SIAM Journal on Optimization*, volume 21, pages 231–268, 2011.

[15] Kenneth L. Clarkson, Elad Hazan, and David P. Woodruff. Sublinear optimization for machine learning. In *Proceedings of the 2010 IEEE 51st Annual Symposium on Foundations of Computer Science*, FOCS '10, pages 449–457, 2010.

[16] Elad Hazan. Approximate convex optimization by online game playing. *CoRR*, abs/cs/0610119, 2006.

[17] Kenneth L. Clarkson, Elad Hazan, and David P. Woodruff. Sublinear optimization for machine learning. *CoRR*, abs/1010.4408, 2010.

[18] Shai Shalev-shwartz, Yoram Singer, and Andrew Y. Ng. Online and batch learning of pseudo-metrics. In *ICML*, pages 743–750, 2004.

[19] James Hardy Wilkinson. *The algebric eigenvalue problem*. Claderon Press, Oxford, 1965.

[20] Gene H. Golub and Charles F. Van Loan. *Matrix computations*. John Hopkins University Press, 1989.

[21] Martin Zinkevich. Online convex programming and generalized infinitesimal gradient ascent. In *ICML*, pages 928–936, 2003.

